# Relative Performance Guarantees for Approximate Inference in Latent Dirichlet Allocation

**Indraneel Mukherjee**　　　　**David M. Blei**

Department of Computer Science
Princeton University
35 Olden Street
Princeton, NJ 08540
{imukherj,blei}@cs.princeton.edu

## Abstract

Hierarchical probabilistic modeling of discrete data has emerged as a powerful tool for text analysis. Posterior inference in such models is intractable, and practitioners rely on approximate posterior inference methods such as variational inference or Gibbs sampling. There has been much research in designing better approximations, but there is yet little theoretical understanding of which of the available techniques are appropriate, and in which data analysis settings. In this paper we provide the beginnings of such understanding. We analyze the improvement that the recently proposed collapsed variational inference (CVB) provides over mean field variational inference (VB) in latent Dirichlet allocation. We prove that the difference in the tightness of the bound on the likelihood of a document decreases as $O(k-1) + \sqrt{\log m/m}$, where $k$ is the number of topics in the model and $m$ is the number of words in a document. As a consequence, the advantage of CVB over VB is lost for long documents but increases with the number of topics. We demonstrate empirically that the theory holds, using simulated text data and two text corpora. We provide practical guidelines for choosing an approximation.

## 1   Introduction

Hierarchical probabilistic models of discrete data have emerged as powerful tool for large-scale text analysis. Based on latent semantic indexing (LSI) [1] and probabilistic latent semantic indexing (pLSI) [2], hierarchical topic models [3, 4] have been extended and applied to sequential settings [5, 6], authorship [7], email [8], social networks [9, 10], computer vision [11, 12], bioinformatics [5, 13], information retrieval [14], and other application areas [15, 16, 17, 18]. See [19] for a good review.

A topic model posits a generative probabilistic process of a document collection using a small number of distributions over words, which are called topics. Conditioned on the observed documents, the distribution of the underlying latent variables is inferred to probabilistically partition the data according to their hidden themes. Research in topic models has involved tailoring the latent structure to new kinds of data and designing new posterior inference algortihms to infer that latent structure.

In generative models, such as latent Dirichlet allocation (LDA) and its extensions, inferring the posterior of the latent variables is intractable [3, 4]. (Some topic models, such as LSI and pLSI, are not fully generative.) Several algorithms have emerged in recent years to approximate such posteriors, including mean-field variational inference [3], expectation propagation [20], collapsed Gibbs sampling [19] and, most recently, collapsed variational inference [21]. Choosing from among the several available algorithms is difficult. There has been some empirical comparison in the topic modeling literature [4, 19], but little theoretical guidance.

We provide some of the first theoretical understanding of which of the available techniques is appropriate, and in which data analysis settings. We analyze two variational inference algorithms for topic models, *mean field variational inference* (VB) [3] and *collapsed variational inference* (CVB) [21]. "Collapsing," or marginalizing out, a latent variable is a known technique for speeding up the convergence of Gibbs samplers, and CVB brought this idea to the world of variational algorithms. Empirically, CVB was more accurate than VB for LDA [21]. The advantage of CVB applied to Dirichlet process mixtures was less conclusive [22].

Variational algorithms minimize the distance between a simple distribution of the latent variables and the true posterior. This is equivalent to maximizing a lower bound on the log probability of a document. We prove that the uncollapsed variational bound on the log probability of a document approaches the collapsed variational bound as the number of words in the document increases. This supports the empirical improvement observed for LDA, where documents are relatively short, and the smaller improvement observed in the DP mixture, which is akin to inference in a single long document. We also show how the number of topics and the sparsity of those topics affects the performance of the two algorithms.

We prove that the difference between the two bounds decreases as $O(k-1) + \sqrt{\log m/m}$, where $k$ is the number of topics in the model, and $m$ is the number of words in the document. Thus, the advantage of CVB over VB is lost for longer documents. We examine the consequences of the theory on both simulated and real text data, exploring the relative advantage of CVB under different document lengths, topic sparsities, and numbers of topics. The consequences of our theory lead to practical guidelines for choosing an appropriate variational algorithm.

## 2    Posterior inference for latent Dirichlet allocation

Latent Dirichlet allocation (LDA) is a model of an observed corpus of documents. Each document is a collection of $m$ words $x_{1:m}$, where each word is from a fixed vocabulary $\chi$ of size $N$. The model parameters are $k$ topics, $\beta_1, \ldots, \beta_k$, each of which is a distribution on $\chi$, and a $k$-vector $\vec{\alpha}$, which is the parameter to a Dirichlet over the $(k-1)$-simplex. The topic matrix $\beta$ denotes the $N \times k$ matrix whose columns are the topic distributions.

Given the topic matrix and Dirichlet parameters, LDA assumes that each document arises from the following process. First, choose topic proportions $\theta \sim \mathcal{D}(\vec{\alpha})$. Then, for each word choose a topic assignment $z_i \sim \theta$. Finally, choose the word $x_i \sim \beta_{z_i}$. This describes a joint probability distribution of the observed and latent variables $p(\vec{x}, \vec{z}, \theta | \vec{\alpha}, \beta)$.

Analyzing data with LDA involves two tasks. In parameter estimation, we find the topics and Dirichlet parameters that maximize the likelihood of an observed corpus. In posterior inference, we fix the model and compute the posterior distribution of the latent structure that underlies a particular document. Here, we focus on posterior inference. (Parameter estimation crucially depends on posterior inference via the expectation-maximization algorithm.)

Given a document $\vec{x}$, the posterior distribution of the latent variables is $p(\theta, \vec{z}|\vec{x}) = \frac{p(\theta, \vec{z}, \vec{x})}{p(\vec{x})}$. This distribution is infeasible to compute exactly because of the difficulty in computing the normalizing constant, i.e., the marginal probability of the document,

$$p(\vec{x}) = \frac{\Gamma(\sum_z \alpha_z)}{\prod_z \Gamma(\alpha_z)} \sum_{\vec{z}} \int \left( \prod_z \theta_z^{\alpha_z - 1} \right) \left( \prod_i \beta_{z_i, x_i} \theta_{z_i} \right) d\theta.$$

Approximating the posterior is equivalent to approximating the normalizing constant.

Variational methods approximate an intractable posterior by finding the member of a simpler family of distributions that is closest to it, where closeness is measured by relative entropy. This is equivalent to minimizing the Jensen's bound on the negative log probability of the data [23]. We will analyze two alternative variational methods.

**Variational inference for LDA**    In the variational inference algorithm for LDA introduced in [3] (VB), the posterior $p(\theta, \vec{z}|\vec{x})$ is approximated by a fully-factorized variational distribution

$$q(\theta, \vec{z}|\vec{\gamma}, \phi_{1:m}) = q(\theta|\vec{\gamma}) \prod_i q(z_i|\phi_i).$$

Here $q(\theta|\vec{\gamma})$ is a Dirichlet distribution with parameters $\vec{\gamma}$, and each $q(z_i|\phi_i)$ is a multinomial distribution on the set of $K$ topic indices. This family does not contain the true posterior. In the true posterior, the latent variables are dependent; in this family of distributions, they are independent [3].

The algorithm seeks to find the variational parameters that minimize the relative entropy between the true posterior and the approximation, $\mathtt{RE}(q(\theta, \vec{z}|\vec{\gamma}, \phi_{1:m}) \parallel p(\theta, \vec{z}|\vec{x}))$. This is equivalent to finding the optimal parameters $\vec{\gamma}_*, \phi_{1:m}^*$ as follows:

$$(\vec{\gamma}_*, \phi_{1:m}^*) = \arg\min_{\vec{\gamma}, \phi_{1:m}} \left[ \mathbb{E}_{q(\theta, \vec{z}|\vec{\gamma}, \phi_{1:m})} \log \left( \frac{q(\theta, \vec{z}|\vec{\gamma}, \phi_{1:m})}{p(\theta, \vec{z}, \vec{x})} \right) \right].$$

The expression minimized by $\vec{\gamma}_*, \phi_{1:m}^*$ is also known as the *variational free energy* of $(\vec{\gamma}, \phi_{1:m})$ and will be denoted by $\mathcal{F}(\vec{x}, \vec{\gamma}, \phi_{1:m})$. Note that $\mathcal{F}(\vec{x}, \vec{\gamma}_*, \phi_{1:m}^*)$ is the Jensen's bound on the negative log probability of $\vec{x}$. The value of the objective function is a measure of the quality of the VB approximation. We denote this with

$$\mathtt{VB}(\vec{x}) \triangleq \min_{\vec{\gamma}, \phi_{1:m}} \mathcal{F}(\vec{x}, \vec{\gamma}, \phi_{1:m}). \tag{1}$$

**Collapsed variational inference for LDA** The collapsed variational inference algorithm (CVB) reformulates the LDA model by marginalizing out the topic proportions $\theta$. This yields a formulation where the topic assignments $z$ are fully dependent, but where the dimensionality of the latent space has been reduced.

The variational family in CVB is a fully-factorized product of multinomial distributions,

$$q(z) = \prod_i q(z_i|\phi_i).$$

CVB finds the optimal variational parameters $\phi_{1:m}^*$ as follows:

$$\phi_{1:m}^* = \arg\min_{\phi_{1:m}} \left[ \mathbb{E}_{q(\vec{z}|\phi_{1:m})} \log \left( \frac{q(\vec{z}|\phi_{1:m})}{p(\vec{z}, \vec{x})} \right) \right].$$

It approximates the negative log probability of $\vec{x}$ with the *collapsed variational free energy* $\mathcal{F}(\vec{x}, \vec{\gamma})$, which is the expression that $\phi_{1:m}^*$ minimizes. Analogous to VB, CVB's performance is measured by

$$\mathtt{CVB}(\vec{x}) \triangleq \min_{\phi_{1:m}} \mathcal{F}(\vec{x}, \phi_{1:m}). \tag{2}$$

Both $\mathtt{CVB}(\vec{x})$ and $\mathtt{VB}(\vec{x})$ approximate the negative log probability of $\vec{x}$ by Jensen's inequality. It has been shown that $\mathtt{CVB}(\vec{x})$ will always be a better bound than $\mathtt{VB}(\vec{x})$ [21].

**Efficiency of the algorithms** Both VB and CVB proceed by coordinate ascent to reach a local minimum of their respective free energies. CVB achieves higher accuracy at the price of increased computation. Each coordinate update for VB requires in $O(mk)$ time, where $m$ is the length of a document and $k$ is the number of topics. Each coordinate update for CVB requires $O(m^2 k)$ time. The CVB updates are prohibitive for large documents and, moreover, are numerically unstable. Both shortcomings are overcome in [21] by substituting exact computations with an efficient second-order Taylor approximation. This approximation, however, does not yield a proper bound.[1] It is thus inappropriate for computing held out probability, a typical measure of quality of a topic model. For such a quantity, exact CVB implementation takes quadratic time.

## 3 Relative performance of VB and CVB

We try to obtain a theoretical handle on the size of the advantage of CVB over VB, and how it is affected by the length of the document, the number of topics, and the structure of those topics. Our main result states that for sufficiently large documents, the difference in approximation quality decreases with document length and converges to a constant that depends on the number of topics.

**Theorem 1.** *Consider any LDA model with $k$ topics, and a document consisting of $m$ words $x_1, \ldots, x_m$, where $m$ is sufficiently large. Recall that $VB(\vec{x})$ and $CVB(\vec{x})$, defined in (1) and (2), are the free energies measured by VB and CVB respectively. Then,*

$$0 \leq [VB(\vec{x}) - CVB(\vec{x})] \leq O(k-1) + o(1) \tag{3}$$

*for this model. Here $o(1)$ goes to $0$ at least as fast as $\sqrt{\frac{\log m}{m}}$.*

A strength of Theorem 1 is that it holds for any document, and not necessarily one generated by an LDA model. In previous work on analyzing mean-field variational inference, [24] analyze the performance of VB for posterior inference in a Gaussian mixture model. Unlike the assumptions in Theorem 1, their analysis requires that the data be generated by a specific model.

Topic models are often evaluated and compared by approximation of the per-word log probability. Concerning this quantity, the following corollary is immediate because the total free energy scales with the length of the document.

**Corollary 1.** *The per word free energy change, as well as the percentage free energy change, between VB and CVB goes to zero with the length of the document.*

Our results are stated in log-space. The bounds on the difference in free energy is equivalent to a bound on the ratio of probability obtained by VB and CVB. Since the probability of a document falls exponentially fast with the number of words, the additive difference in the probability estimates of VB and CVB is again negligible for large documents.

**Corollary 2.** *For sufficiently long documents, the difference in probability estimates of CVB and VB decrease as $c^{m-k}$ for some constant $c < 1$ whose value depends on the model parameters $\beta$.*

The upper-bound in (3) is nearly tight. When all topics are uniform distributions, the difference in the free energy estimates is $\Omega(k)$ for long documents.

## 3.1 Proof Sketch

We sketch the proof of Theorem 1. The full proof is in the supporting material. We first introduce some notation. We denote a vector with an arrow, like $\vec{\nu}$. All vectors have $k$ real coordinates. $\nu_j$ will denote its coordinates, with $j \in [k] = \{1, \ldots, k\}$. When iterating over indices in $[k]$, we will use the variable $j$. To iterate from $1$ to $m$ we will use $i$.

We state three lemmas which are needed to prove (3). The left inequality in (3) follows from the fact that CVB optimizes over a larger family of distributions [21]. We concentrate on the right inequality. The first step is to carry out calculations similar to [24] to arrive at the following.

**Lemma 1.** *Suppose $q(\vec{z}) = \prod_i q_i(z_i)$ is the optimal approximation to the posterior $p(\vec{z}|\vec{x})$. Then,*

$$VB(\vec{x}) - CVB(\vec{x}) \leq \sum_z \left( \mathbb{E}_{q(\vec{z})}[\log \Gamma(m_j + \alpha_j)] - \log \Gamma(\gamma_j + \alpha_j) \right) \tag{4}$$

*where $\gamma_j = \sum_i q_i(Z_i = j), \forall j \in [k]$, and $m_j$ is the number of occurrences of the topic $j$ in $\vec{z}$.*

Note that to analyze the term $\mathbb{E}_{q(\vec{z})}[\log \Gamma(m_j + \alpha_j)]$ corresponding to a particular topic $j$, we need consider only those positions $i$ where $q_i(Z_i = j) \neq 0$; we denote the number of such positions by $N_z$. The difficulty in analyzing arbitrary documents lay in working with the right hand side of (4) without any prior knowledge about the $q_i$'s. This was overcome by the following lemma.

**Lemma 2.** *Suppose $X_i$ is Bernoulli random with probability $q_i$, for $i = 1$ to $m$. Let $f : \mathbb{R} \to \mathbb{R}$ be convex, and $\gamma \in [0, m]$. Then the following optimization problem is solved when each $q_i = \frac{\gamma}{m}$*

$$\begin{aligned} \max_{q_1, \ldots, q_m} \quad & \mathbb{E}[f(X_1 + \ldots + X_m)] \\ s.t. \quad & q_i \in [0, 1] \\ & q_1 + \ldots + q_m = \gamma. \end{aligned}$$

As an immediate corollary of the previous two lemmas and the fact that $\log \Gamma$ is convex, we get

$$VB(\vec{x}) - CVB(\vec{x}) \leq \sum_j \mathbb{E}[\log \Gamma(m_j + \alpha_j)] - \log \Gamma(\gamma_j + \alpha_j).$$

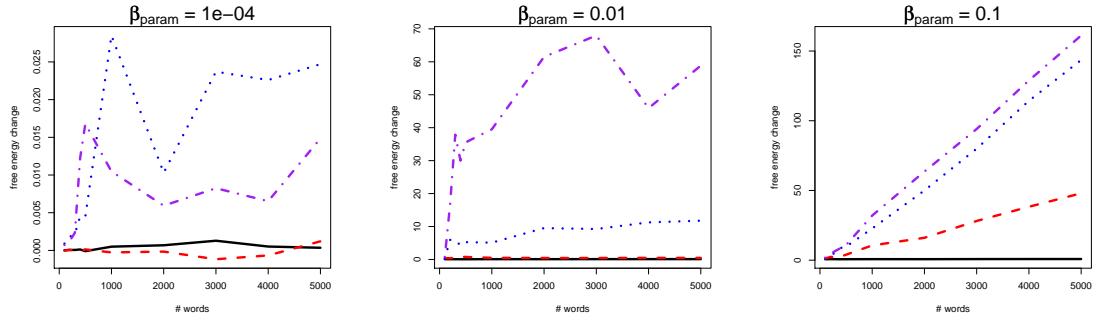

(a) Difference in total free energy estimates

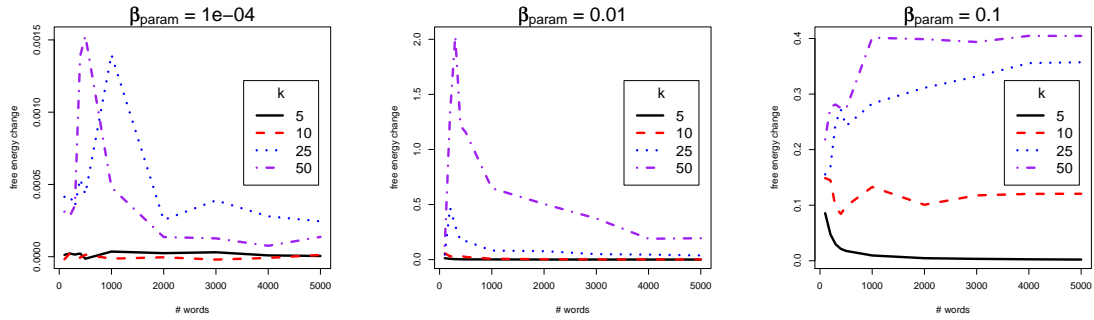

(b) Percentage difference in free energy estimates

Figure 1: Results on synthetic text data. We sample $k$ topics from a symmetric Dirichlet distribution with parameter $\beta_{\text{param}}$. We sample 10 documents from LDA models with these topics. We consider prefixes of varying lengths for each document. For each prefix length, the VB and CVB free energies are averaged over the 10 documents. The curves obtained show how the advantage of CVB over VB changes with the length of a document, number of topics and sparsity of topics.

where $m_j$ is now a Binomial random variable with probability $\frac{\gamma_j}{m}$ and number of trials $m$. The last piece of the proof is the following concentration lemma.

**Lemma 3.** *Let $X$ be the number of heads in $m$ coin tosses each with probability $q$. We require $m > q^{-(2+o(1))}$. Let $a > 0$ be constants. Then*

$$0 \le \mathbb{E}[\log \Gamma(X + a)] - \log \Gamma(\mathbb{E}[X + a]) \le O(1 - q) + o(1) \tag{5}$$

*Here $o(1) = O(\sqrt{\frac{\log m}{m}})$.*

The requirement of $m > 1/q^{2+o(1)}$ is necessary, and translates to the condition that document lengths be greater than $(N_j/\gamma_j)^{2+o(1)}$ for Theorem 1 to hold. This gives an implicit lower bound on the required length of a document which depends on the sparsity of the topics. (Sparse topics place their mass on few words, i.e., low entropy, and dense topics spread their mass on more words, i.e., high entropy). When the vocabulary is large, dense topics require long documents for the theory to take effect. This is supported by our simulations.

## 4 Empirical results

We studied the results of this theory on synthetic and real text data. We implemented the algorithms described in [3] and [21]. While these algorithms are only guaranteed to find a local optimum of the objective, we aim to study whether our theorem about the global optimum is borne out in practice.

**Synthetic data**    The synthetic data was generated as follows. We first sampled $k$ topics $\beta_1, \ldots, \beta_k$ independently from a symmetric Dirichlet distribution with parameter $\beta_{\text{param}}$. We then sampled a corpus of 10 documents, each of length 5000 from an LDA model with these topics and Dirichlet hyper-parameter $1/k$. The vocabulary size was 10000.

For each document, we considered sub-documents of the first $m$ words with lengths as small as 100. On each sub-document, we ran both VB and CVB initialized from a common point. For every sub-document length, the average converged values of the free energy was recorded for both algorithms. Thus, we obtain a trajectory representing how the advantage of CVB over VB changes with the number of words $m$.

We repeated this simulation with different values of $k$ to reveal the dependence of this advantage on the number of topics. Moreover, we investigated the dependence of the advantage on topic sparsity. We repeat the above experiment, with three different values of the Dirichlet parameter $\beta_{\text{param}}$ for the topic matrix. The topics become sparse rapidly as $\beta_{\text{param}}$ decreases.

The results of this study are in Figure 1. We see similar trends across all data. The advantage decreases with document length $m$ and increases with the number of topics $k$. The theory predicts that the difference in free energy converges to a constant, implying that the percentage advantage decays as $O(1)/m$. Figure 1 reveals this phenomenon. Moreover, the constant is estimated to be on the order of $k$, implying that the advantage is higher for more topics. Comparing the curves for different values of $k$ reveals this fact. Finally, for denser topic models the performances of CVB and VB converge only for very long documents, as was discussed at the end of Section 3.1. When $\beta_{\text{param}} = 0.1$, CVB retains its advantage even for 5000 word long documents.

**Real-world corpora**    We studied the relative performance of the algorithms on two text data sets. First, we examined 3800 abstracts from the ArXiv, an on-line repository of scientific pre-prints. We restricted attention to 5000 vocabulary terms, removing very frequent and very infrequent terms. Second, we examined 1000 full documents from the Yale Law Journal. Again, we used a vocabulary of 5000 terms. Each data set was split into a training and test corpus. The ArXiv test corpus contained 2000 short documents. The Yale Law test corpus contained 200 documents of lengths between a thousand and $10,000$ words.

For each data set, we fit LDA models of different numbers of topics to the training corpus ($k = 5, 10, 25, 50$), and then evaluated the model on the held-out test set. In Figure 2, we plot the percentage difference of the per-word variational free energies achieved by `CVB` and `VB` as a function of document length and number of topics. We also plot the difference in the total free energy. As for the simulated data, the graphs match our theory; the percent decrease in per word free energy goes to zero with increasing document length, and the absolute difference approaches a constant. The difference is more pronounced as the number of topics increases.

The predicted trends occur even for short documents containing around a hundred words. Topics estimated from real-world data tend to be sparse. The issues seen with dense topics on simulated data are not relevant for real-world applications.

## 5    Conclusion

We have provided a theoretical analysis of the relative performance of the two variational inference algorithms for LDA. We showed that the advantage of CVB decreases as document length increases, and increases with the number of topics and density of the topic distributions. Our simulations on synthetic and real-world data empirically confirm our theoretical bounds and their consequences. Unlike previous analyses of variational methods, our theorem does not require that the observed data arise from the assumed model.

Since the approximation to the likelihood based on CVB is more expensive to compute than for VB, this theory can inform our choice of a good variational approximation. Shorter documents and models with more topics lend themselves to analysis with CVB. Longer documents and models with fewer topics lend themselves to VB. One might use both, within the same data set, depending on the length of the document.

Figure 2: Experiments with the two text data sets described in Section 4. We fit LDA models with numbers of topics equal to $5, 10, 25, 50$, and evaluated the models on a held-out corpus. We plot the percentage difference of the per-word variational free energies achieved by CVB and VB as a function of document length. We also plot the difference in the total free energy. The %-age decrease in per word free energy goes to zero with increasing document length, and the absolute difference approaches a constant. The difference is higher for larger $k$.

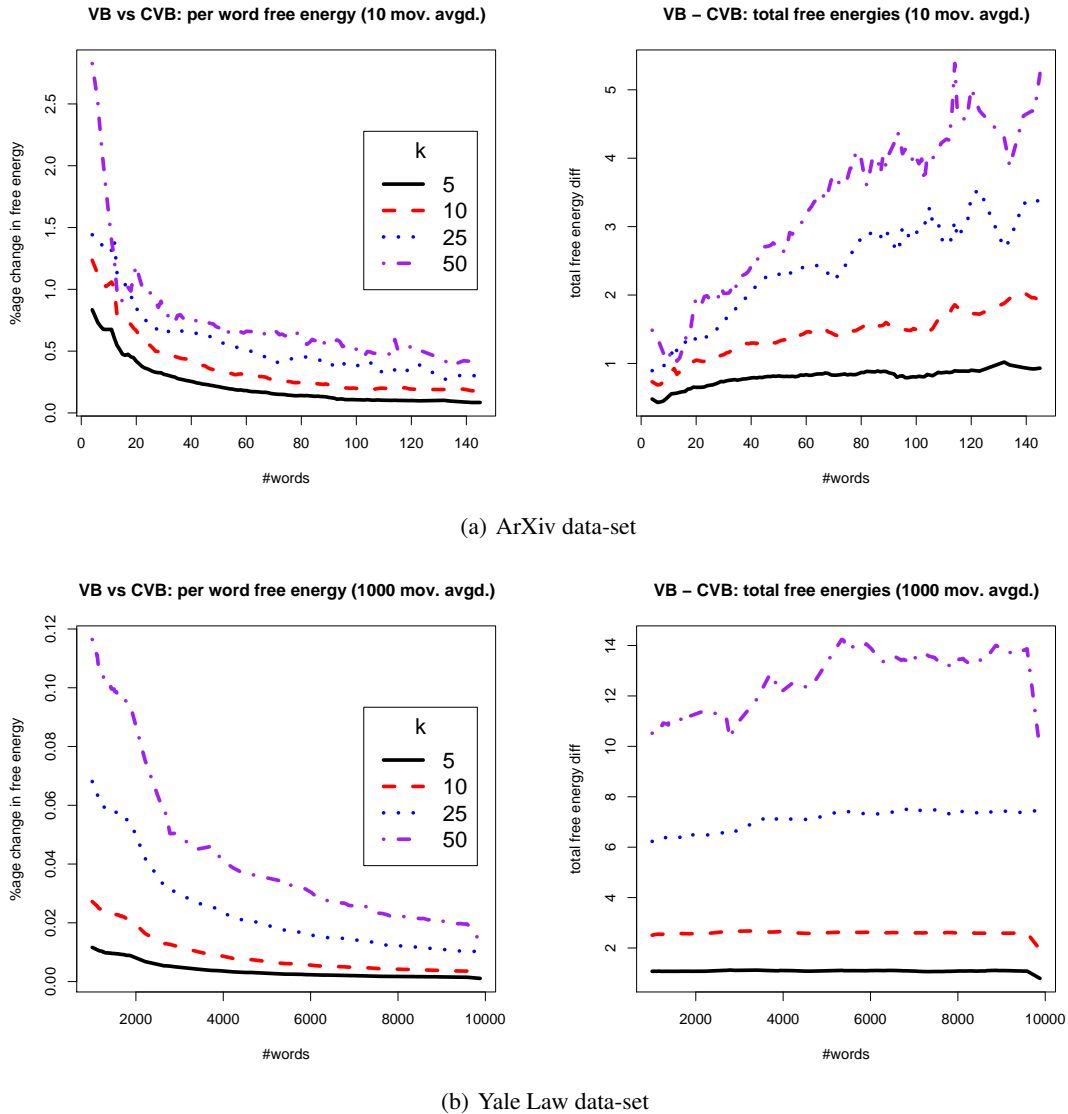

(a) ArXiv data-set

(b) Yale Law data-set

In one strain of future work, we will analyze the consequences of the approximate posterior inference algorithm on parameter estimation. Our results regarding the sparsity of topics indicate that CVB is a better algorithm early in the EM algorithm, when topics are dense, and that VB will be more efficient as the fitted topics become more sparse.

## Footnotes

[1]The first-order Taylor approximation yields an upper-bound, but these turn out to be too inaccurate. Such an estimate can yield bounds worse than those achieved by VB.

# References

[1] S. Deerwester, S. Dumais, T. Landauer, G. Furnas, and R. Harshman. Indexing by latent semantic analysis. *Journal of the American Society of Information Science*, 41(6):391–407, 1990.

[2] T. Hofmann. Probabilistic latent semantic analysis. In *UAI*, 1999.

[3] D. Blei, A. Ng, and M. Jordan. Latent Dirichlet allocation. *Journal of Machine Learning Research*, 3:993–1022, 2003.

[4] W. Buntine and A. Jakulin. Discrete component analysis. In *Subspace, Latent Structure and Feature Selection*. Springer, 2006.

[5] M. Girolami and A. Kaban. Simplicial mixtures of Markov chains: Distributed modelling of dynamic user profiles. In *NIPS 16*, pages 9–16. MIT Press, 2004.

[6] H. Wallach. Topic modeling: Beyond bag of words. In *Proceedings of the 23rd International Conference on Machine Learning*, 2006.

[7] M. Rosen-Zvi, T. Griffiths, M. Steyvers, and P. Smith. The author-topic model for authors and documents. In *Proceedings of the 20th Conference on Uncertainty in Artificial Intelligence*, pages 487–494. AUAI Press, 2004.

[8] A. McCallum, A. Corrada-Emmanuel, and X. Wang. The author-recipient-topic model for topic and role discovery in social networks: Experiments with Enron and academic email. Technical report, University of Massachusetts, Amherst, 2004.

[9] E. Airoldi, D. Blei, S. Fienberg, and E. Xing. Mixed membership stochastic blockmodels. *arXiv*, May 2007.

[10] D. Zhou, E. Manavoglu, J. Li, C. Giles, and H. Zha. Probabilistic models for discovering e-communities. In *WWW Conference*, pages 173–182, 2006.

[11] L. Fei-Fei and P. Perona. A Bayesian hierarchical model for learning natural scene categories. *IEEE Computer Vision and Pattern Recognition*, pages 524–531, 2005.

[12] B. Russell, A. Efros, J. Sivic, W. Freeman, and A. Zisserman. Using multiple segmentations to discover objects and their extent in image collections. In *IEEE Conference on Computer Vision and Pattern Recognition*, pages 1605–1614, 2006.

[13] S. Rogers, M. Girolami, C. Campbell, and R. Breitling. The latent process decomposition of cDNA microarray data sets. *IEEE/ACM Transactions on Computational Biology and Bioinformatics*, 2(2):143–156, 2005.

[14] X. Wei and B. Croft. LDA-based document models for ad-hoc retrieval. In *SIGIR*, 2006.

[15] D. Mimno and A. McCallum. Organizing the OCA: Learning faceted subjects from a library of digital books. In *Joint Conference on Digital Libraries*, 2007.

[16] B. Marlin. Collaborative filtering: A machine learning perspective. Master's thesis, University of Toronto, 2004.

[17] C. Chemudugunta, P. Smyth, and M. Steyvers. Modeling general and specific aspects of documents with a probabilistic topic model. In *NIPS 19*, 2006.

[18] D. Andrzejewski, A. Mulhern, B. Liblit, and X. Zhu. Statistical debugging using latent topic models. In *European Conference on Machine Learning*, 2007.

[19] T. Griffiths and M. Steyvers. Probabilistic topic models. In T. Landauer, D. McNamara, S. Dennis, and W. Kintsch, editors, *Latent Semantic Analysis: A Road to Meaning*. Laurence Erlbaum, 2006.

[20] T. Minka and J. Lafferty. Expectation-propagation for the generative aspect model. In *Uncertainty in Artificial Intelligence (UAI)*, 2002.

[21] Y. Teh, D. Newman, and M. Welling. A collapsed variational bayesian inference algorithm for latent dirichlet allocation. In *NIPS*, pages 1353–1360, 2006.

[22] K. Kurihara, M. Welling, and Y. Teh. Collapsed variational Dirichlet process mixture models. 2007.

[23] M. Jordan, Z. Ghahramani, T. Jaakkola, and L. Saul. Introduction to variational methods for graphical models. *Machine Learning*, 37:183–233, 1999.

[24] K. Watanabe and S. Watanabe. Stochastic complexities of gaussian mixtures in variational bayesian approximation. *Journal of Machine Learning Research*, 7:625–644, 2006.

